# A "Shape Aware" Model for semi-supervised Learning of Objects and its Context

**Abhinav Gupta[1], Jianbo Shi[2] and Larry S. Davis[1]**
[1] Dept. of Computer Science, Univ. of Maryland, College Park
[2] Dept. of Computer and Information Sciences, Univ. of Pennsylvania
agupta@cs.umd.edu, jshi@cis.upenn.edu, lsd@cs.umd.edu

## Abstract

We present an approach that combines bag-of-words and spatial models to perform semantic and syntactic analysis for recognition of an object based on its internal appearance and its context. We argue that while object recognition requires modeling relative spatial locations of image features within the object, a bag-of-word is sufficient for representing context. Learning such a model from weakly labeled data involves labeling of features into two classes: foreground(object) or "informative" background(context). We present a "shape-aware" model which utilizes contour information for efficient and accurate labeling of features in the image. Our approach iterates between an MCMC-based labeling and contour based labeling of features to integrate co-occurrence of features and shape similarity.

## 1 Introduction

Understanding the meaning of a sentence involves both syntactic and semantic analysis. A bag-of-words approach applied locally over a sentence would be insufficient to understand its meaning. For example, "Jack hit the bar" and "The bar hit Jack" have different meanings even though the bag-of-words representation is the same for both. In many cases, determining meaning also requires word sense disambiguation using contextual knowledge. For example, does "bar" represents a rod or a place where drinks are served? While a combined semantic and syntactical model could be used for representation and application of context as well, it would be expensive to apply. Syntactical rules are generally not required for extracting knowledge about context - a topic model is generally sufficient for contextual analysis in text [14, 15].

We use analogous reasoning to suggest a similar dichotomy in representing object structure and context in vision. Our approach combines bag-of-words and spatial models to capture semantics and syntactic rules, respectively, that are employed for recognizing an object using its appearance, structure and context. We treat an object and a scene analogous to a sentence and a document respectively. Similar to documents, object recognition in natural scenes requires modeling spatial relationships of image features(words) within the object but for representing context in a scene, a bag-of-words approach suffices (See Figure 1 (a) and (b)).

Learning such a model from weakly labeled data requires labeling the features in an image as belonging to an object or its context (informative background). Spatial models, such as constellation or star models, compute a sparse representation of objects(with a fixed number of parts) by selecting features which satisfy spatial constraints. Their sparse representation reduces their utility in the presence of occlusion. Approaches for learning a dense bag-of-features model with spatial constraints from weakly labeled data have also been proposed. Such approaches (based on marginalizing over possible locations of the object), however, lead to poor foreground segmentation if the training dataset is small, the images have significant clutter [1] or if some other object in the background has a strong and consistent spatial relationship with the object to be learned throughout the

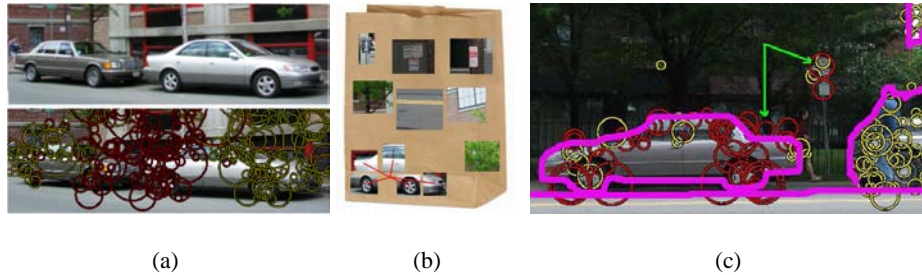

(a)                              (b)                              (c)

Figure 1: (a) An example of the importance of spatial constraints locally. The red color shows the features on the foreground car. A bag of words approach fails to capture spatial structure and thus combines the front and rear of different cars. (b) We use a spatial model of the object and a bag-of-words approach for context representation. (c) Importance of using contour information: Objects such as signs become part of the foreground since they occur at consistent relative location to the car. If shape and contour information is combined with co-occurrence and spatial structure of image features, then such mis-labellings can be reduced. For example, in the above case since there are strong intervening contours between the features on the car(foreground) and the features on signs, and there is a lack of strong contours between features on signs and features on trees (background), it is more likely that features on the signs should be labeled as background.

---

***Problem:***
Learn the parameters of object model given the images $(I_1, .., I_D)$, object labels $(O_1, .., O_D)$
and Object Model Shape $(M)$.

***Approach:***
Simultaneous localization the object in training images and estimation of model parameters. This
is achieved by integrating cues from image features and contours. The criteria includes following terms:
*1.* **Feature Statistics:** *The image features satisfy the co-occurrence and spatial statistics of the model.*
*2.* **Shape Similarity:** *The shape of the foreground object is similar to the shape of the sketch of the object.*
*3.* **Separation:** *The object and background features should be separated by the object boundary contours.*

---

Table 1: Summary of "Shape Aware" Model

training dataset. We overcome this problem by applying shape based constraints while constructing the foreground model.

Figure 1(c) shows an example of how contours provide important information for foreground/background labeling. We add two constraints to the labeling problem using the contour information: (a) The first constraint requires the presence of strong intervening contours between foreground and background features. (b) The second constraint requires the shape of boundary contours be similar to the shape of the exemplar/sketch provided with the weakly labeled dataset. This allows us to learn object models from images where there is significant clutter and in which the object does not cover a significant part of the image. We provide an iterative solution to integrate these constraints. Our approach first labels the image features based on co-occurrence and spatial statistics - the features that occur in positive images and exhibit strong spatial relationships are labeled as foreground features. Based on the labels of image features, object boundaries are identified based on how well they separate foreground and background features. This is followed by a shape matching step which identifies the object boundary contours based on their expected shape. This step prunes many contours and provides a better estimate of object boundaries. These boundaries are then be used to relabel the features in the image. This provides an initialization point for the next iteration of Gibbs sampling. Figure 2 shows the system flow of our "Shape Aware" approach.

## 1.1   Related Work

Many graphical models for object recognition [11] have been inspired by models of text documents such as LDA [6] and pLSA [7]. These models are computationally efficient because they ignore the spatial relationships amongst image features (or parts) and use a dense object representation. However, ignoring spatial relationships between features leads to problems (See Figure 1(a)). In contrast, approaches that model spatial relationships [9, 5] between object parts/features are com-

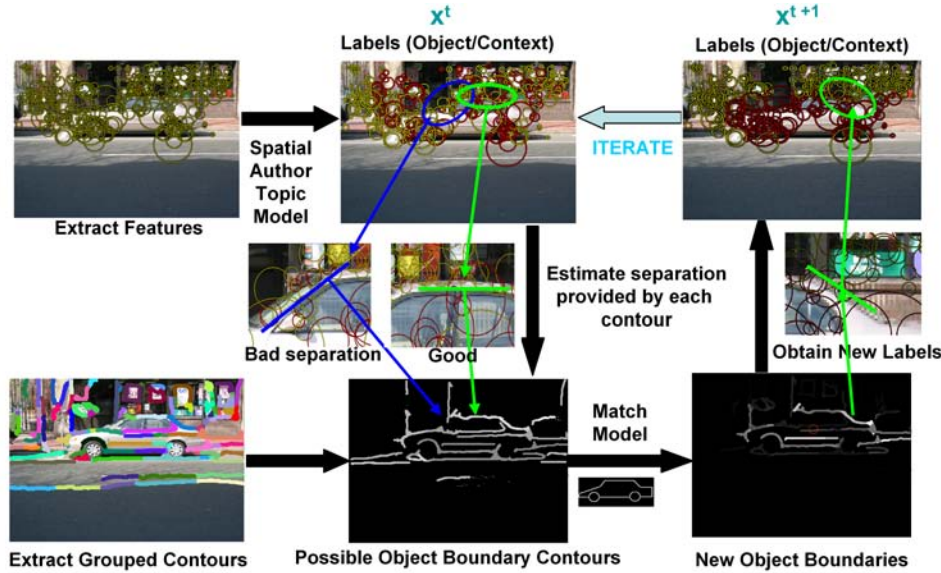

Figure 2: Shape-Aware Learning (Overview): We first compute feature labels using the Gibbs sampling approach on the Spatial Author Topic model. The features labeled foreground and background are drawn in red and yellow respectively. This is followed by object boundary extraction. The object boundaries are identified based on how well they separate foreground and background features. Likely object boundary contours are then matched to the sketch using a voting-based approach and the contours consistent with the shape of the sketch are identified. These contours are then used to relabel the features using the same separation principle. The new labels and topics from the previous time step are used as a new initialization point for the next iteration.

putationally expensive and therefore employ only sparse features representation. These approaches fail under occlusion due to their sparse representation and their stringent requirement of a one-one correspondence between image and object features.

There has been recent work in applying spatial constraints to topic models which enforce neighboring features to belong to similar topics [10, 2] for the purpose of segmentation. Our work is more related to classification based approaches [8, 3] that model spatial locations of detected features based on a reference location in the image. Sudderth et. al [3] presented such a model that can be learned in a supervised manner. Fergus et. al [8] proposed an approach to learn the model from weakly labeled data. This was achieved by marginalizing object locations and scale. Each object location hypothesis provides a foreground segmentation which can be used for learning the model. Such an approach, however, is expensive unless the training images are not highly cluttered. Additionally, they are subject to modeling errors if the object of interest is small in the training images.

Our goal is to simultaneously learn an object model and its context model from weakly labeled images. To learn context we require real world scenes of object and their natural surrounding environment (high clutter and small objects). We present a "shape aware" feature based model for recognizing objects. Our approach resolves the foreground/background labeling ambiguities by requiring that the shapes of the foreground object across the training images to be similar to a sketch exemplar. Shape based models [1] have been used previously for object recognition. However, contour matching is an expensive(exponential) problem due to the need to select the best subset of contours from the set of all edges that match the shape model. Approximate approaches such as MCMC are not applicable since matching is very closely coupled with selection. We propose an efficient approach that iterates between an co-occurence based labeling and contour based labeling of features.

## 2   Our Approach - Integrating feature and contour based cues

We assume the availability of a database of weakly labeled images which specify the presence of an object, but not its location. Similar to previous approaches based on document models, we vector

quantize the space of image features into visual words to generate a discrete image representation. Each visual word is analogous to a word and an image is treated analogous to a document.

Each word is associated with a topic and an author (the object). The topic distribution depends on the associated author and the word distribution depends on the assigned topic (Section 2.1). We start with random assignments of words to topics and authors. This is followed by a Gibbs sampling step which simultaneously estimates the hidden variables (topic and author) and also the parameters of the generative model that maximizes the likelihood(Section 2.2). These assignments are then used to obtain a set of likely object boundary contours in each image. These contours are subsequently analyzed to identify the object "centers" and final object contours by matching with the shape exemplar(Section 2.3). Using the new set of boundary contours, the authors corresponding to each word are reassigned and the model is retrained using the new assignment.

## 2.1 Generative Model - Syntax and Semantics

**Author-Topic Model:** Our model is motivated by the author-topic model [13] and the model presented in [4]. We first provide a brief description of the author topic model, shown in figure 3(a). The author-topic model is used to model documents for which a set of authors is given. For each word in the document, an author ($x_i$) is chosen uniformly at random from the set of authors ($a_d$). A topic ($z_i$) is chosen from a distribution of topics specific to the selected author and a word ($w_i$) is generated from that topic. The distribution of topics ($\theta$) for each author is chosen from a symmetric Dirichlet($\alpha$) prior and the distribution of words ($\phi$) for a topic is chosen from symmetric Dirichlet ($\beta$) prior.

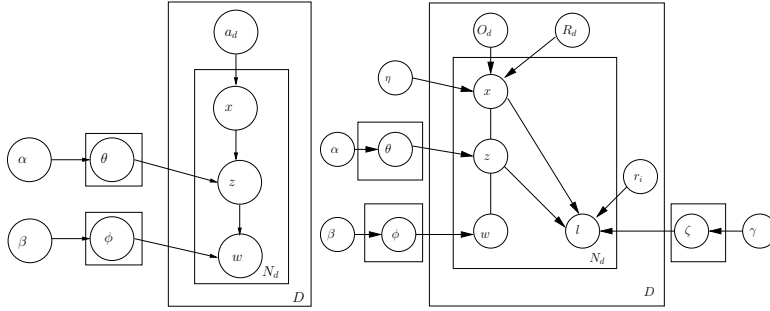

Figure 3: (a) Author-Topic Model (b) Our Model (Spatial Author-Topic Model). Our model extends the author topic model by including the spatial(syntactical) relationship between features.

**Spatial-Author Topic Model:** Our model is shown in figure 3(b). Our goal is not only to model the distribution of type of features but also to model the distribution of spatial locations of the subset of these features that are associated with the foreground object. We model this as follows: A feature in the image is described by its type $w_i$ and location $l_i$. Each feature ($w_i, l_i$) is 'authored' by an author $x_i$ which is described by its type $o_i$ [2] and its location $r_i$. For each feature, the author $x_i$ is chosen from a distribution, $\eta$, which can be either uniform or generated using available priors from other sources. Topic $z_i$ for each word is chosen from a distribution of topic specific to the type of object $o_i$ and a word $w_i$ is generated from that topic. The distribution of topics ($\theta$) for each object type is chosen from a symmetric Dirichlet ($\alpha$) distribution [3]. The distribution of a word for each topic is chosen from a symmetric Dirichlet ($\beta$) prior.

The location of each feature, $l_i$, is sampled from the distribution $p(l_i|o_i, z_i, r_i)$ using the following distribution:

$$p(l_i|o_i, z_i, r_i) = exp(\frac{-||l_i - r_i||^2}{\sigma_s^2})\zeta_{r_i}^{o_i, z_i}(l_i) \tag{1}$$

The first term ensures that each feature has higher probability of being generated by nearby reference locations. The second term enforces spatial constraints on the location of the feature that is generated by topic ($z_i$). We enforce these spatial constraints by a binning approach. Each feature in the foreground can lie in $B$ possible bins with respect to the reference location. The distribution of the spatial location of a feature is specific to the topic $z_i$ and the type of object $o_i$. This distribution is chosen from a symmetric Dirichlet ($\gamma$) prior. Since we do not want to enforce spatial constraints on the locations of the features generated by topics from context, we set $\zeta$ to a constant when $o_i$ corresponds to the context of some object.

## 2.2 Gibbs Sampling

We use Gibbs sampling to estimate $z_i$ and $x_i$ for each feature. Given the features $(w, l)$, authors assignments $x$, other topic assignments $z_{-i}$ and other hyperparameters, each $z_i$ is drawn from:

$$
\begin{aligned}
P(z_i|w, l, x, z_{-i}) &\propto P(w_i|w_{-i}, z)P(z_i|z_{-i}, o_i)P(l_i|x_i, l_{-i}, x_{-i}, z_i) \\
&\propto \frac{n_{w_i}^{z_i} + \beta}{n^{z_i} + W\beta} \frac{n_{z_i}^{o_i} + \alpha}{n^{o_i} + T\alpha} \frac{n_{B_i}^{o_i, z_i} + \gamma}{n^{o_i, z_i} + B\gamma}
\end{aligned}
\tag{2}
$$

where $n_{w_i}^{z_i}$ represents the number of features of type $w_i$ in the dataset assigned to topic $z_i$, $n^{z_i}$ represents the total number of features assigned to topic $z_i$. $n_{z_i}^{o_i}$ represents the number of features that are assigned to topic $z_i$ and author of type $o_i$ and $n^{o_i}$ represents the total number of features assigned to author $o_i$. $B_i$ represents the spatial bin in which feature $i$ lies in when the reference is $r_i$, $n_{B_i}^{o_i, z_i}$ represents the number of features from object type $o_i$ and topic $z_i$ which lie in bin $B_i$, $n^{o_i, z_i}$ represents the total number of features from object type $o_i$ and topic $z_i$. $W$ is number of type of words and $T$ represents number of topic types.

Similarly, given the features $(w, l)$, topic assignments $z$, other author assignments $x_{-i}$ and other hyperparameters, each $x_i$ is drawn from:

$$
\begin{aligned}
P(x_i|w, l, z, x_{-i}) &\propto P(l_i|x_i, l_{-i}, x_{-i}, z_i)P(z_i|o_i, z_{-i}, x_{-i})P(r_i|o_i, z_{-i}, x_{-i}) \\
&\propto exp(\frac{-||l_i - r_i||^2}{\sigma_s^2})\frac{n_{B_i}^{o_i, z_i} + \gamma}{n^{o_i, z_i} + B\gamma} \frac{n_{z_i}^{o_i} + \alpha}{n^{o_i} + T\alpha} \frac{n_{r_i}^{o_i} + \delta}{n^{o_i} + R\delta}
\end{aligned}
\tag{3}
$$

where $n_{r_i}^{o_i}$ represents the number of features from object type $o_i$ that have $r_i$ as the reference location and $n^{o_i}$ represents the total number of features from object $o_i$. In case $o_i$ is of type context, the second term is replaced by a constant. $R$ represents the number of possible reference locations.

## 2.3 "Shape Aware" Model

The generative model presented in section 2.1 can be learned using the Gibbs sampling approach explained above. However, this approach has some shortcomings: (a) If there are features in the background that exhibit a strong spatial relationship with the object, they can be labeled as foreground. (b) In clutter, the labeling performance diminishes as the discriminability of the object is lower. The labeling performance can, however, be improved if contour cues are utilized. We do this by requiring that the shape of the object boundary contours extracted based on feature labeling should be similar to a sketch of the object provided in the dataset. Thus, the labeling of features into foreground and background is not only governed by co-occurrence and structural information, but also by shape similarity. We refer to this as a "shape aware" model.

Shape matching using contours has, in the worst case, exponential complexity since it requires selection of the subset of contours that best constitute the foreground boundary. We avoid this computationally expensive challenge by solving the selection problem based on the labels of features extracted using Gibbs sampling. The spatial author-topic model is used to attend to the contours which are likely to be object boundaries. Our shape matching module has three steps: (a) Extracting object boundaries based on labels extracted from the spatial author topic model. (b) Extracting boundaries consistent with the shape model by matching. (c) Using new boundaries to determine new labels for features.

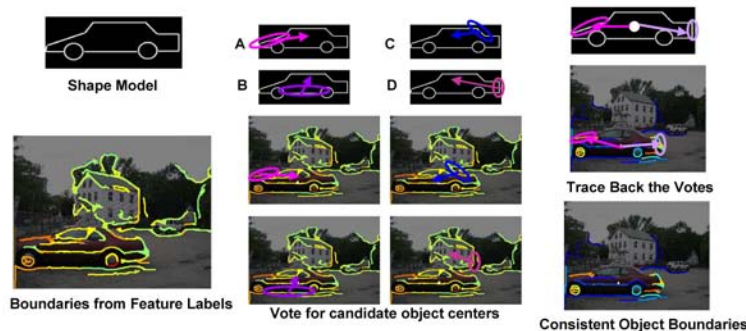

Figure 4: Extraction of object boundaries consistent with the shape of exemplar. The first step is extraction of contours which separate foreground and background features. This is followed by a voting process. Each contour in the image is matched to every contour in the model to extract the center of the object. The votes are then traced back to identify the contours consistent with the shape model.

**Extracting Object Boundary Contours from Feature Labels:** We first determine the edges using and group them into contours using the approach presented in [16]. Each contour $c_j$ is a collection of 2D points $(p_{j1}, p_{j2}....)$. Our goal is to extract boundary contours of the object using the feature labels. Since, the boundary contours separates foreground and background features, an estimate of the number of foreground and background features on each side of an image contour provides evidence as to whether that image contour is part of the object boundary. For each contour, we measure the number of foreground and background features that lie on each side of the contour within some fixed distance of the contour. The probability that a contour is a boundary contour $cl_j = 1$ of the object with the side $S1$ being the interior of the object is given by:

$$P_{S1}(cl_j = 1|x) = \frac{n_f^{S1} + \tau}{n^{S1} + 2\tau} \frac{n_b^{S2} + \tau}{n^{S2} + 2\tau} \qquad (4)$$

where $n_f^{S1}$ is the total number of features with foreground label on side $S1$ of the contour and $n^{S1}$ is total number of features on side $S1$.

**Shape Matching:** Given the probabilities of each contour being a part of the object boundary, we estimate the object center using a voting-based approach [18]. Each contour votes for the center of the object where the weight of the vote is determined based on how well the contour matches the sketch. Non-maximal suppression is then used to estimate the candidate object locations. Once the candidate location of the center of object is selected, we trace back the votes to estimate the new boundary of the object. Figure 4 shows an example of the voting process and boundary contours extracted using this approach.

**Extracting New Labels:** These boundaries are then used to relabel the image features into foreground and background. We use the same separation principle to label new features. Each boundary contour votes as to whether a feature should be labeled foreground or background. If the feature lies on the same side as the object center, then the contour votes for the feature as foreground. Votes are weighted based on the probability of a contour being an object boundary. Therefore, the probability that the feature $i$ is labeled as foreground is given by $\frac{\sum_j \omega_j \nu_{ij}}{\sum_j \omega_j}$ where $\omega_j$ is the probability that the contour $j$ is on object boundary and $\nu_{ij}$ is variable which is 1 if the object center and feature are on same side of contour $c_j$ or 0, if the center is on opposite side. The new labels are then used as an initialization point for the Gibbs sampling based learning of the feature model.

## 3 Experimental Results

We tested our "shape-aware" model on images of cars obtained from the Label-me dataset[17]. We randomly selected 45 images for training the model from the LabelMe dataset. A potential concern is the number of iterations/convergence required by our iterative approach. However, it was empirically observed that, in most cases the system stabilizes after only two iterations. It should also be noted that each iteration between contour and feature labelings is performed after 200 iterations

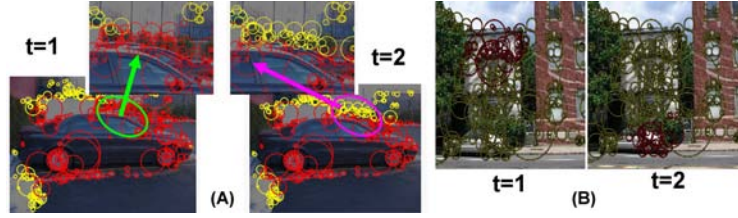

Figure 5: Advantages of iterative approach. At each iteration, the author topic distribution changes, which requires retraining the model using Gibbs sampling. This can help in two ways: (A) More Focused Attention: The feature labeling gets refined. (B) Change of Focus: A new reference point gets chosen by new distribution.

of Gibbs sampling. The advantages of having an iterative approach is shown in, figure 5. We compared the performance of our system against the author-topic model and the author-topic model with spatial constraints. We evaluated the performance of the algorithm by measuring the labeling performance in training and test datasets. Better labeling in training is required for better model learning. Figure 6 show some of the cases where both author-topic and author-topic model with spatial constraints fail due to high clutter or the foreground object being too small in the training dataset. The "shape aware" model, however, shows better localization performance as compared to the other two.

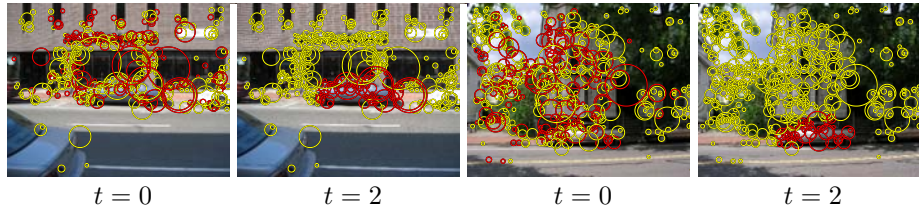

$t = 0$ $\qquad$ $t = 2$ $\qquad$ $t = 0$ $\qquad$ $t = 2$

Figure 6: Two examples of how the "shape aware" model provides better localization compared to spatial author topic models. The odd columns show the results of the author topic model (the initialization point of iterative approach). The even columns show the labeling provided by our algorithm after 2 iterations.

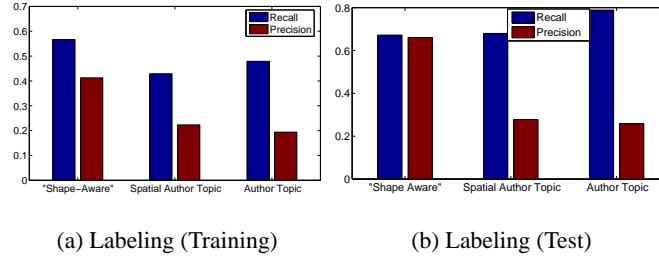

(a) Labeling (Training) $\qquad\qquad$ (b) Labeling (Test)

Figure 7: Quantitative Comparison of author-topic, spatial author-topic and "shape aware" model based on randomly selected 40 images each from the training and test dataset(17000 features each approximately). The values of the parameters used are $T = 50$, $\alpha = \frac{50}{T}$, $\beta = 0.01$, $\gamma = 0.01$, $B = 8$ and $\tau = 0.1$.

Figure 7 shows a quantitative comparison of the "shape aware" model to the author-topic and the spatial author-topic model. Recall ratio is defined as the ratio of features labeled as foreground to the total number of foreground features. Precision is defined as the ratio of features correctly labeled as foreground to the total number of features labeled as foreground. In the case of labeling in training data, our approach outperforms both author-topic and spatial author-topic model. In the case of test dataset, the author-topic model has higher recall but very low precision. The low precision of author-topic and spatial author-topic can be attributed to the fact that, in many cases the context is similar and at the same relative locations to each other. This leads to modeling errors - these features are learned to be part of the object. In the case of the "shape aware" model, the shape of the objects help in pruning these features and therefore lead to much higher precision. Low recall rates in our model and the spatial author-topic model is because some foreground features do not satisfy the spatial

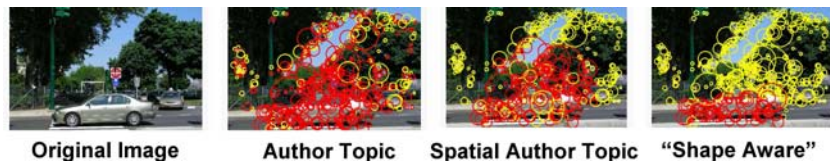

**Original Image**     **Author Topic**     **Spatial Author Topic**     **"Shape Aware"**

Figure 8: Example of performance of three models on a test image. "Shape Aware" model shows high precision in label prediction due to pruning provided by shape matching. Author Topic model shows high recall rates because high similarity in context across images.

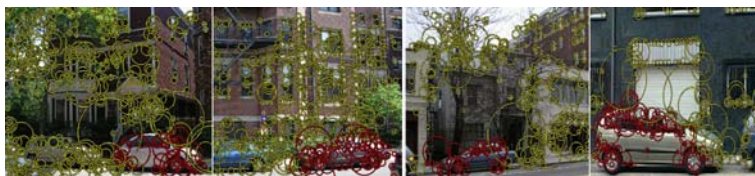

Figure 9: A few examples of labeling in the test dataset.

constraints and hence are falsely labeled as background features. Figure 9 shows some examples of performance of the "shape aware" model on test dataset.

## Acknowledgements

This research was funded by US Government's VACE program and NSF-IIS-04-47953(CAREER) award. The authors would also like to thank Qihui Zhu for providing the code for extracting contours.

## Footnotes

[1] A dataset of less cluttered images would fail to provide enough contextual information to be learned for a model that simultaneously learns object model and its contextual relationships.

[2] For an image with label car, the possible object types are car, and context of car. The differentiation between "informative" and "non-informative" background is captured by the probability distributions.

[3] The Dirichlet distribution is an attractive distribution - it belongs to the exponential family and is conjugate to the multinomial distribution.

## References

[1] G. Elidan, G. Heitz and D. Koller, *Learning Object Shape: From Drawings to Images*, IEEE CVPR 2006.

[2] X. Wang and E. Grimson, *Spatial Latent Dirichlet Allocation*, NIPS 2007.

[3] E. Sudderth, A. Torralba, W.T Freeman and A.S Wilsky, *Learning Hierarchical Models of Scenes, Objects and Parts*, ICCV 2005.

[4] T.L Griffiths, M Steyvers, D.M Blei and J.B Tenenbaum, *Integrating Topics and Syntax*, NIPS 2005.

[5] D.J Crandall and D.P Huttenlocher, *Weakly Supervised Learning of Part-Based Spatial Models for Visual Object Recognition*, ECCV 2006.

[6] D. Blei, A. Ng and M. Jordan, *Latent Dirichlet Allocation*, Journal of Machine Learning Research, 2003.

[7] T. Hofmann, *Unsupervised learning by probabilistic latent semantic analysis*, Machine Learning 2001.

[8] R. Fergus, L. Fei-Fei, P. Perona and A. Zisserman, *Learning Object Categories from Google's Image Search*, ICCV 2005.

[9] R. Fergus, P. Perona and A. Zisserman, *Object Class Recognition by Unsupervised Scale-Invariant Learning*, CVPR 2003.

[10] L. Cao and L. Fei-Fei, *Spatially coherent latent topic model for concurrent object segmentation and classification*, ICCV 2007.

[11] B. Russell, A. Efros, J. Sivic, W. Freeman and A. Zisserman, *Using Multiple Segmentations to Discover Objects and their Extent in Image Collections*, CVPR 2006.

[12] T.L Griffiths and M. Steyvers, *Finding Scientific Topics*, PNAS 2004.

[13] M. Rosen-Zvi, T. Griffiths, M. Steyvers and P. Smyth, *The Author-Topic Model for Authors and Documents*, UAI 2004

[14] M. Lesk, *Automatic Sense Disambiguation Using Marchine Readable Dictionaries: How to Tell a Pine Cone from Ice Cream Cone*, SIGDOC 1986.

[15] D. Yarowsky, *Word Sense Disambiguation Using Statistical Models of Roget's Categories trained on Large Corpora*, COLING 1992.

[16] Q. Zhi, G. Song and J. Shi, *Untangling Cycles for Contour Grouping*, ICCV 2007.

[17] B. C. Russell, A. Torralba, K. P. Murphy, W. T. Freeman, *LabelMe: a Database and Web-based Tool for Image Annotation*, IJCV 2008.

[18] B. Leibe, A. Leonardis and B. Schiele,*Combined Object Categorization and Segmentationwith an Implicit Shape Model*, ECCV workshop on Statistical Learning in Vision, 2006.
